# Maximum Conditional Likelihood via Bound Maximization and the CEM Algorithm

**Tony Jebara and Alex Pentland**
Vision and Modeling, MIT Media Laboratory, Cambridge MA
http://www.media.mit.edu/ ~ jebara
{ jebara,sandy }@media.mit.edu

## Abstract

We present the CEM (*Conditional Expectation Maximization*) algorithm as an extension of the EM (*Expectation Maximization*) algorithm to conditional density estimation under missing data. A bounding and maximization process is given to specifically optimize conditional likelihood instead of the usual joint likelihood. We apply the method to conditioned mixture models and use bounding techniques to derive the model's update rules. Monotonic convergence, computational efficiency and regression results superior to EM are demonstrated.

## 1 Introduction

Conditional densities have played an important role in statistics and their merits over joint density models have been debated. Advantages in feature selection, robustness and limited resource allocation have been studied. Ultimately, tasks such as regression and classification reduce to the evaluation of a conditional density.

However, popularity of maximum joint likelihood and EM techniques remains strong in part due to their elegance and convergence properties. Thus, many conditional problems are solved by first estimating joint models then conditioning them. This results in concise solutions such as the Nadarya-Watson estimator [2], Xu's mixture of experts [7], and Amari's em-neural networks [1]. However, direct conditional density approaches [2, 4] can offer solutions with higher conditional likelihood on test data than their joint counter-parts.

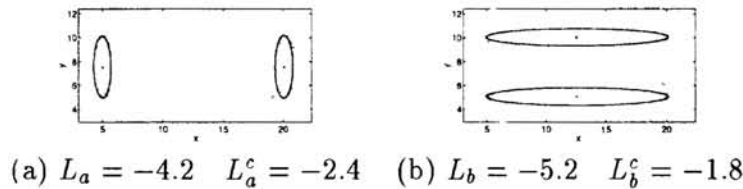

(a) $L_a = -4.2$  $L_a^c = -2.4$  (b) $L_b = -5.2$  $L_b^c = -1.8$

Figure 1: Average Joint $(x, y)$ vs. Conditional $(y|x)$ Likelihood Visualization

Popat [6] describes a simple visualization example where 4 clusters must be fit with 2 Gaussian models as in Figure 1. Here, the model in (a) has a superior joint likelihood ($L_a > L_b$) and hence a better $p(x, y)$ solution. However, when the models are conditioned to estimate $p(y|x)$, model (b) is superior ($L_b^c > L_a^c$). Model (a) yields a poor unimodal conditional density in $y$ and (b) yields a bi-modal conditional density. It is therefore of interest to directly optimize conditional models using conditional likelihood. We introduce the CEM (*Conditional Expectation Maximization*) algorithm for this purpose and apply it to the case of Gaussian mixture models.

## 2   EM and Conditional Likelihood

For joint densities, the tried and true EM algorithm [3] maximizes joint likelihood over data. However, EM is not as useful when applied to conditional density estimation and maximum conditional likelihood problems. Here, one typically resorts to other local optimization techniques such as gradient descent or second order Hessian methods [2]. We therefore introduce CEM, a variant of EM, which targets conditional likelihood while maintaining desirable convergence properties. The CEM algorithm operates by directly bounding and decoupling conditional likelihood and simplifies M-step calculations.

In EM, a complex density optimization is broken down into a two-step iteration using the notion of missing data. The unknown data components are estimated via the E-step and a simplified maximization over complete data is done in the M-step. In more practical terms, EM is a bound maximization: the E-step finds a lower bound for the likelihood and the M-step maximizes the bound.

$$p(\mathbf{x}_i, \mathbf{y}_i | \Theta) = \sum_{m=1}^{M} p(m, \mathbf{x}_i, \mathbf{y}_i | \Theta) \tag{1}$$

Consider a complex joint density $p(\mathbf{x}_i, \mathbf{y}_i | \Theta)$ which is best described by a discrete (or continuous) summation of simpler models (Equation 1). Summation is over the 'missing components' $m$.

$$
\begin{aligned}
\Delta l &= \sum_{i=1}^{N} \log(p(\mathbf{x}_i, \mathbf{y}_i | \Theta^t)) - \log(p(\mathbf{x}_i, \mathbf{y}_i | \Theta^{t-1})) \\
&\geq \sum_{i=1}^{N} \sum_{m=1}^{M} h_{im} \log \frac{p(m, \mathbf{x}_i, \mathbf{y}_i | \Theta^t)}{p(m, \mathbf{x}_i, \mathbf{y}_i | \Theta^{t-1})} \quad \text{where} \quad h_{im} = \frac{p(m, \mathbf{x}_i, \mathbf{y}_i | \Theta^{t-1})}{\sum_{n=1}^{M} p(n, \mathbf{x}_i, \mathbf{y}_i | \Theta^{t-1})}
\end{aligned}
\tag{2}
$$

By appealing to Jensen's inequality, EM obtains a lower bound for the incremental log-likelihood over a data set (Equation 2). Jensen's inequality bounds the logarithm of the sum and the result is that the logarithm is applied to each simple

model $p(m, \mathbf{x}_i, \mathbf{y}_i | \Theta)$ individually. It then becomes straightforward to compute the derivatives with respect to $\Theta$ and set to zero for maximization (M-step).

$$p(\mathbf{y}_i | \mathbf{x}_i, \Theta) = \sum_{m=1}^{M} p(m, \mathbf{y}_i | \mathbf{x}_i, \Theta) = \frac{\sum_{m=1}^{M} p(m, \mathbf{x}_i, \mathbf{y}_i | \Theta)}{\sum_{m=1}^{M} p(m, \mathbf{x}_i | \Theta)} \quad (3)$$

However, the elegance of EM is compromised when we consider a conditioned density as in Equation 3. The corresponding incremental conditional log-likelihood, $\Delta l^c$, is shown in Equation 4.

$$
\begin{aligned}
\Delta l^c &= \sum_{i=1}^{N} \log(p(\mathbf{y}_i | \mathbf{x}_i, \Theta^t)) - \log(p(\mathbf{y}_i | \mathbf{x}_i, \Theta^{t-1})) \\
&= \sum_{i=1}^{N} \log \frac{\sum_{m=1}^{M} p(m, \mathbf{x}_i, \mathbf{y}_i | \Theta^t)}{\sum_{m=1}^{M} p(m, \mathbf{x}_i, \mathbf{y}_i | \Theta^{t-1})} - \log \frac{\sum_{n=1}^{M} p(n, \mathbf{x}_i | \Theta^t)}{\sum_{n=1}^{M} p(n, \mathbf{x}_i | \Theta^{t-1})}
\end{aligned}
\quad (4)
$$

The above is a difference between a ratio of joints *and* a ratio of marginals. If Jensen's inequality is applied to the second term in Equation 4 it yields an *upper* bound since the term is subtracted (this would compromise convergence). Thus, only the first ratio can be lower bounded with Jensen (Equation 5).

$$\Delta l^c \geq \sum_{i=1}^{N} \sum_{m=1}^{M} h_{im} \log \frac{p(m, \mathbf{x}_i, \mathbf{y}_i | \Theta^t)}{p(m, \mathbf{x}_i, \mathbf{y}_i | \Theta^{t-1})} - \log \frac{\sum_{n=1}^{M} p(n, \mathbf{x}_i | \Theta^t)}{\sum_{n=1}^{M} p(n, \mathbf{x}_i | \Theta^{t-1})} \quad (5)$$

Note the lingering logarithm of a sum which prevents a simple M-Step. At this point, one would resort to a Generalized EM (GEM) approach which requires gradient or second-order ascent techniques for the M-step. For example, Jordan *et al.* overcome the difficult M-step caused by EM with an Iteratively Re-Weighted Least Squares algorithm in the mixtures of experts architecture [4].

## 3   Conditional Expectation Maximization

The EM algorithm can be extended by substituting Jensen's inequality for a different bound. Consider the upper variational bound of a logarithm $x - 1 \geq \log(x)$ (which becomes a lower bound on the negative log). The proposed logarithm's bound satisfies a number of desiderata: (1) it makes contact at the current operating point[1], (2) it is tangential to the logarithm, (3) it is a tight bound, (4) it is simple and (5) it is the variational dual of the logarithm. Substituting this linear bound into the incremental conditional log-likelihood maintains a true lower bounding function $Q$ (Equation 6).

$$\Delta l^c \geq Q(\Theta^t, \Theta^{t-1}) = \sum_{i=1}^{N} \sum_{m=1}^{M} h_{im} \log \frac{p(m, \mathbf{x}_i, \mathbf{y}_i | \Theta^t)}{p(m, \mathbf{x}_i, \mathbf{y}_i | \Theta^{t-1})} - \frac{\sum_{n=1}^{M} p(n, \mathbf{x}_i | \Theta^t)}{\sum_{n=1}^{M} p(n, \mathbf{x}_i | \Theta^{t-1})} + 1 \quad (6)$$

The Mixture of Experts formalism [4] offers a graceful representation of a conditional density using experts (conditional sub-models) and gates (marginal sub-models). The $Q$ function adopts this form in Equation 7.

$$\sum_{i=1}^{N}\sum_{m=1}^{M}\left\{h_{im}(\log p(\mathbf{y}_i|m,\mathbf{x}_i,\Theta^t)+\log p(m,\mathbf{x}_i|\Theta^t)-z_{im})-r_ip(m,\mathbf{x}_i|\Theta)+\tfrac{1}{M}\right\}$$
$$\text{where} \quad z_{im}=\log(p(m,\mathbf{x}_i,\mathbf{y}_i|\Theta^{t-1})) \quad \text{and} \quad r_i=\left(\sum_{n=1}^{M}p(n,\mathbf{x}_i|\Theta^{t-1})\right)^{-1} \tag{7}$$

Computing this $Q$ function forms the CE-step in the Conditional Expectation Maximization algorithm and it results in a simplified M-step. Note the absence of the logarithm of a sum and the *decoupled* models. The form here allows a more straightforward computation of derivatives with respect to $\Theta^t$ and a more tractable M-Step. For continuous missing data, a similar derivation holds.

At this point, without loss of generality, we specifically attend to the case of a conditioned Gaussian mixture model and derive the corresponding M-Step calculations. This serves as an implementation example for comparison purposes.

## 4 CEM and Bound Maximization for Gaussian Mixtures

In deriving an efficient M-step for the mixture of Gaussians, we call upon more bounding techniques that follow the CE-step and provide a monotonically convergent learning algorithm. The form of the conditional model we will train is obtained by conditioning a joint mixture of Gaussians. We write the conditional density in a experts-gates form as in Equation 8. We use unnormalized Gaussian gates $\bar{\mathcal{N}}(\mathbf{x};\mu,\Sigma)=\exp(-\tfrac{1}{2}(\mathbf{x}-\mu)^T\Sigma^{-1}(\mathbf{x}-\mu))$ since conditional models do not require true marginal densities over $\mathbf{x}$ (i.e. that necessarily integrate to 1). Also, note that the parameters of the gates $(\alpha,\mu_x,\Sigma_{xx})$ are independent of the parameters of the experts $(\nu^m,\Gamma^m,\Omega^m)$.

Both gates and experts are optimized independently and have no variables in common. An update is performed over the experts and then over the gates. If each of those causes an increase, we converge to a local maximum of conditional log-likelihood (as in Expectation Conditional Maximization [5]).

$$
\begin{aligned}
p(\mathbf{y}|\mathbf{x},\Theta) &= \frac{\sum_{m=1}^{M}\alpha_n\bar{\mathcal{N}}(\mathbf{X};\mu_x^n,\Sigma_{xx}^n)\times\mathcal{N}(\mathbf{y};\mu_y^m+\Sigma_{yx}^m(\Sigma_{xx}^m)^{-1}(\mathbf{X}-\mu_x^m),\Sigma_{yy}^m-\Sigma_{yx}^m(\Sigma_{xx}^m)^{-1}\Sigma_{xy}^m)}{\sum_{n=1}^{M}\alpha_n\bar{\mathcal{N}}(\mathbf{X};\mu_x^n,\Sigma_{xx}^n)} \\
&= \frac{\sum_{m=1}^{M}\alpha_n\bar{\mathcal{N}}(\mathbf{X};\mu_x^n,\Sigma_{xx}^n)\times\mathcal{N}(\mathbf{y};\nu^m+\Gamma^m\mathbf{X},\Omega^m)}{\sum_{n=1}^{M}\alpha_n\bar{\mathcal{N}}(\mathbf{X};\mu_x^n,\Sigma_{xx}^n)}
\end{aligned}
\tag{8}
$$

To update the experts, we hold the gates fixed and merely take derivatives of the $Q$ function with respect to the expert parameters ($\Phi^m=\{\nu^m,\Gamma^m,\Omega^m\}$) and set them to 0. Each expert is effectively decoupled from other terms (gates, other experts, etc.). The solution reduces to maximizing the log of a single conditioned Gaussian and is analytically straightforward.

$$\frac{\partial Q(\Theta^t,\Theta^{(t-1)})}{\partial\Phi^m} = \sum_{i=1}^{N}h_{im}\frac{\partial\log\mathcal{N}(\mathbf{y}_i;\nu^m+\Gamma^m\mathbf{x}_i,\Omega^m)}{\partial\Phi^m} := 0 \tag{9}$$

Similarly, to update the gate mixing proportions, derivatives of the $Q$ function are taken with respect to $\alpha_m$ and set to 0. By holding the other parameters fixed, the update equation for the mixing proportions is numerically evaluated (Equation 10).

$$\alpha_m := \sum_{i=1}^{N}r_i\hat{\mathcal{N}}(\mathbf{x}_i;\mu_x^m,\Sigma_{xx}^m)\,|_{\Theta^{(t-1)}}\,\{\sum_{i=1}^{N}\hat{h}_{im}\}^{-1} \tag{10}$$

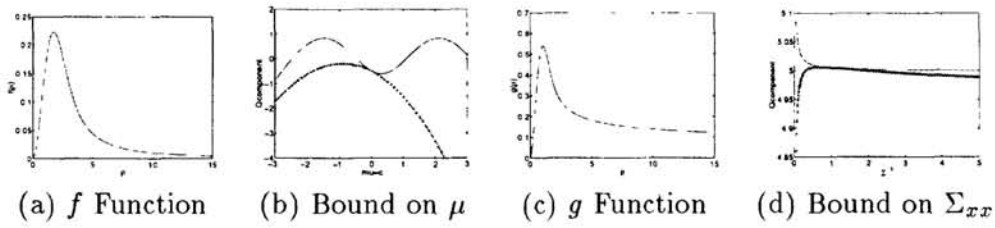

| (a) $f$ Function | (b) Bound on $\mu$ | (c) $g$ Function | (d) Bound on $\Sigma_{xx}$ |

Figure 2: Bound Width Computation and Example Bounds

## 4.1  Bounding Gate Means

Taking derivatives of $Q$ and setting to 0 is not as straightforward for the case of the gate means (even though they are decoupled). What is desired is a simple update rule (i.e. computing an empirical mean). Therefore, we further bound the $Q$ function for the M-step. The $Q$ function is actually a summation of sub-elements $Q_{im}$ and we bound it instead by a summation of quadratic functions on the means (Equation 11).

$$Q(\Theta^t, \Theta^{(t-1)}) = \sum_{i=1}^{N} \sum_{m=1}^{M} Q(\Theta^t, \Theta^{(t-1)})_{im} \geq \sum_{i=1}^{N} \sum_{m=1}^{M} k_{im} - w_{im}\|\mu_x^m - c_{im}\|^2 \quad (11)$$

Each quadratic bound has a location parameter $c_{im}$ (a centroid), a scale parameter $w_{im}$ (narrowness), and a peak value at $k_{im}$. The sum of quadratic bounds makes contact with the $Q$ function at the old values of the model $\Theta^{t-1}$ where the gate mean was originally $\mu_x^{m*}$ and the covariance is $\Sigma_{xx}^{m*}$. To facilitate the derivation, one may assume that the previous mean was zero and the covariance was identity if the data is appropriately whitened with respect to a given gate.

The parameters of each quadratic bound are solved by ensuring that it contacts the corresponding $Q_{im}$ function at $\Theta^{t-1}$ and they have equal derivatives at contact (i.e. tangential contact). Solving these constraints yields quadratic parameters for each gate $m$ and data point $i$ in Equation 12 ($k_{im}$ is omitted for brevity).

$$\begin{aligned} c_{im} &= \frac{1}{2w_{im}}(\hat{h}_{im} - r_i\alpha_m e^{-\frac{1}{2}\mathbf{x}_i^T\mathbf{x}_i})\mathbf{x} \\ w_{im} &\geq r_i\alpha_m \frac{e^{-\frac{1}{2}(\mathbf{x}_i - \mu_x^m)^T(\mathbf{x}_i - \mu_x^m)} - e^{-\frac{1}{2}\mathbf{x}_i^T\mathbf{x}_i} - e^{-\frac{1}{2}\mathbf{x}_i^T\mathbf{x}_i}\mathbf{x}_i^T\mu_x^m}{\mu_x^{m^T}\mu_x^m} + \frac{\hat{h}_{im}}{2} \end{aligned} \quad (12)$$

The tightest quadratic bound occurs when $w_{im}$ is minimal (without violating the inequality). The expression for $w_{im}$ reduces to finding the minimal value, $w_{im}^*$, as in Equation 13 (here $\rho^2 = \mathbf{x}_i^T\mathbf{x}_i$). The $f$ function is computed numerically only *once* and stored as a lookup table (see Figure 2(a)). We thus immediately compute the optimal $w_{im}^*$ and the rest of the quadratic bound's parameters obtaining bounds as in Figure 2(b) where a $Q_{im}$ is lower bounded.

$$w_{im}^* = r_i\alpha_m \overset{\max}{c} \left\{ e^{-\frac{1}{2}\rho^2} \frac{e^{-\frac{1}{2}c^2}e^{c\rho} - c\rho - 1}{c^2} \right\} + \frac{\hat{h}_{im}}{2} = r_i\alpha_m e^{-\frac{1}{2}\rho^2}f(\rho) + \frac{\hat{h}_{im}}{2} \quad (13)$$

The gate means $\mu_x^m$ are solved by maximizing the sum of the $M \times N$ parabolas which bound $Q$. The update is $\mu_x^m = \left(\sum w_{im}^* c_{im}\right)\left(\sum w_{im}^*\right)^{-1}$. This mean is subsequently unwhitened to undo earlier data transformations.

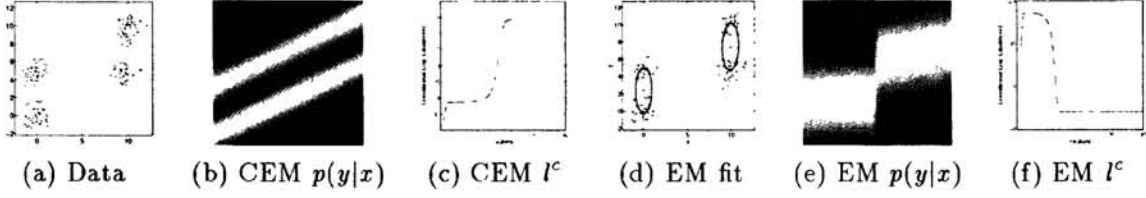

(a) Data    (b) CEM $p(y|x)$    (c) CEM $l^c$    (d) EM fit    (e) EM $p(y|x)$    (f) EM $l^c$

Figure 3: Conditional Density Estimation for CEM and EM

## 4.2 Bounding Gate Covariances

Having derived the update equation for gate means, we now turn our attention to the gate covariances. We bound the $Q$ function with logarithms of Gaussians. Maximizing this bound (a sum of log-Gaussians) reduces to the maximum-likelihood estimation of a covariance matrix. The bound for a $Q_{im}$ sub-component is shown in Equation 14. Once again, we assume the data has been appropriately whitened with respect to the gate's previous parameters (the gate's previous mean is 0 and previous covariance is identity). Equation 15 solves for the log-Gaussian parameters (again $\rho^2 = \mathbf{x}_i^T \mathbf{x}_i$).

$$Q(\Theta^t, \Theta^{(t-1)})_{im} \geq \log(\mathcal{N}) = k_{im} - w_{im}\mathbf{c}_{im}^T {\Sigma_{xx}^m}^{-1} \mathbf{c}_{im} - w_{im}\log|\Sigma_{xx}^m| \quad (14)$$

$$\mathbf{c}_{im}\mathbf{c}_{im}^T = \frac{1}{2w_{im}}\left(\hat{h}_{im} - r_i\alpha_m e^{-\frac{1}{2}\rho^2}\right)\mathbf{x}_i\mathbf{x}_i^T + I$$

$$w_{im} \geq r_i\alpha_m \frac{\frac{1}{2}\exp(-\frac{1}{2}\rho^2)\rho^2 - \frac{1}{2}\exp(-\frac{1}{2}\rho^2)\mathbf{x}_i^T\Sigma^{-1}\mathbf{x}_i + \exp(-\frac{1}{2}\rho^2) - \exp(-\frac{1}{2}\mathbf{x}_i^T\Sigma^{-1}\mathbf{x}_i)}{tr(I) - tr(\Sigma^{-1}) + \log|\Sigma^{-1}|} \quad (15)$$

The computation for the minimal $w_{im}$ simplifies to $w_{im}^* = r_i\alpha_m g(\rho)$. The $g$ function is derived and plotted in Figure 2(c). An example of a log-Gaussian bound is shown in Figure 2(d) a sub-component of the $Q$ function. Each sub-component corresponds to a single data point as we vary one gate's covariance. All $M \times N$ log-Gaussian bounds are computed (one for each data point and gate combination) and are summed to bound the $Q$ function in its entirety.

To obtain a final answer for the update of the gate covariances $\Sigma_{xx}^m$ we simply maximize the sum of log Gaussians (parametrized by $w_{im}^*, k_{im}, \mathbf{c}_{im}$). The update is $\Sigma_{xx}^m = \left(\sum w_{im}^*\mathbf{c}_{im}\mathbf{c}_{im}^T\right)\left(\sum w_{im}^*\right)^{-1}$. This covariance is subsequently unwhitened, inverting the whitening transform applied to the data.

## 5 Results

The CEM algorithm updates the conditioned mixture of Gaussians by computing $h_{im}$ and $r_{im}$ in the CE steps and interlaces these with updates on the experts, mixing proportions, gate means and gate covariances. For the mixture of Gaussians, each CEM update has a computation time that is comparable with that of an EM update (even for high dimensions). However, conditional likelihood (not joint) is monotonically increased.

Consider the 4-cluster $(x, y)$ data in Figure 3(a). The data is modeled with a conditional density $p(y|x)$ using *only 2* Gaussian models. Estimating the density with CEM yields the $p(y|x)$ shown in Figure 3(b). CEM exhibits monotonic conditional likelihood growth (Figure 3(c)) and obtains a more conditionally likely model. In

| Algorithm | CCN0 | CCN5 | C4.5 | LD | EM2 | CEM2 |
|---|---|---|---|---|---|---|
| Abalone | 24.86% | 26.25% | 21.5% | 0.0% | 22.32% | 26.63% |

Table 1: Test Results. Class label regression accuracy data. (CNN0=cascade-correlation, 0 hidden units, CCN5=5 hidden LD=linear discriminant).

the EM case, a joint $p(x, y)$ clusters the data as in Figure 3(d). Conditioning it yields the $p(y|x)$ in Figure 3(e). Figure 3(f) depicts EM's non-monotonic evolution of conditional log-likelihood. EM produces a superior joint likelihood but an inferior conditional likelihood. Note how the CEM algorithm utilized limited resources to capture the multimodal nature of the distribution in $y$ and ignored spurious bimodal clustering in the $x$ feature space. These properties are critical for a good conditional density $p(y|x)$.

For comparison, standard databases were used from UCI [2]. Mixture models were trained with EM and CEM, maximizing joint and conditional likelihood respectively. Regression results are shown in Table 1. CEM exhibited, monotonic conditional log-likelihood growth and out-performed other methods including EM with the same 2-Gaussian model (EM2 and CEM2).

## 6   Discussion

We have demonstrated a variant of EM called CEM which optimizes conditional likelihood efficiently and monotonically. The application of CEM and bound maximization to a mixture of Gaussians exhibited promising results and better regression than EM. In other work, a MAP framework with various priors and a deterministic annealing approach have been formulated. Applications of the CEM algorithm to non-linear regressor experts and hidden Markov models are currently being investigated. Nevertheless, many applications CEM remain to be explored and hopefully others will be motivated to extend the initial results.

### Acknowledgements

Many thanks to Michael Jordan and Kris Popat for insightful discussions.

## Footnotes

[1] The current operating point is 1 since the $\Theta^t$ model in the ratio is held fixed at the previous iteration's value $\Theta^{t-1}$.

[2]http://www.ics.uci.edu/~mlearn/MLRepository.html

## References

[1] S. Amari. Information geometry of em and em algorithms for neural networks. *Neural Networks*, 8(9), 1995.
[2] C. Bishop. *Neural Networks for Pattern Recognition*. Oxford Press, 1996.
[3] A. Dempster, N. Laird, and D. Rubin. Maximum likelihood from incomplete data via the em algorithm. *Journal of the Royal Statistical Society*, B39, 1977.
[4] M. Jordan and R. Jacobs. Hierarchical mixtures of experts and the em algorithm. *Neural Computation*, 6:181–214, 1994.
[5] X. Meng and D. Rubin. Maximum likelihood estimation via the ecm algorithm: A general framework. *Biometrika*, 80(2), 1993.
[6] A. Popat. Conjoint probabilistic subband modeling (phd. thesis). Technical Report 461, M.I.T. Media Laboratory, 1997.
[7] L. Xu, M. Jordan, and G. Hinton. An alternative model for mixtures of experts. In *Neural Information Processing Systems 7*, 1995.

